# EFFICIENT PARALLEL LEARNING ALGORITHMS FOR NEURAL NETWORKS

Alan H. Kramer and A. Sangiovanni-Vincentelli
Department of EECS
U.C. Berkeley
Berkeley, CA 94720

## ABSTRACT

Parallelizable optimization techniques are applied to the problem of learning in feedforward neural networks. In addition to having superior convergence properties, optimization techniques such as the Polak-Ribiere method are also significantly more efficient than the Back-propagation algorithm. These results are based on experiments performed on small boolean learning problems and the noisy real-valued learning problem of hand-written character recognition.

## 1 INTRODUCTION

The problem of learning in feedforward neural networks has received a great deal of attention recently because of the ability of these networks to represent seemingly complex mappings in an efficient parallel architecture. This learning problem can be characterized as an optimization problem, but it is unique in several respects. Function evaluation is very expensive. However, because the underlying network is parallel in nature, this evaluation is easily parallelizable. In this paper, we describe the network learning problem in a numerical framework and investigate parallel algorithms for its solution. Specifically, we compare the performance of several parallelizable optimization techniques to the standard Back-propagation algorithm. Experimental results show the clear superiority of the numerical techniques.

## 2 NEURAL NETWORKS

A neural network is characterized by its architecture, its node functions, and its interconnection weights. In a learning problem, the first two of these are fixed, so that the weight values are the only free parameters in the system. when we talk about "weight space" we refer to the parameter space defined by the weights in a network, thus a "weight vector" $\mathbf{w}$ is a vector or a point in weightspace which defines the values of each weight in the network. We will usually index the components of a weight vector as $w_{ij}$, meaning the weight value on the connection from unit $i$ to unit $j$. Thus $\mathbf{N}(\mathbf{w}, \mathbf{r})$, a network function with $n$ output units, is an $n$-dimensional vector-valued function defined for any weight vector $\mathbf{w}$ and any input vector $\mathbf{r}$:

$$\mathbf{N}(\mathbf{w}, \mathbf{r}) = [o_1(\mathbf{w}, \mathbf{r}), o_2(\mathbf{w}, \mathbf{r}), \ldots, o_n(\mathbf{w}, \mathbf{r})]^T$$

where $o_i$ is the $i_{th}$ output unit of the network. Any node $j$ in the network has input $i_j(\mathbf{w}, \mathbf{r}) = \sum_{i \in \text{fanin}_j} o_i(\mathbf{w}, \mathbf{r}) w_{ij}$ and output $o_j(\mathbf{w}, \mathbf{r}) = f_j(i_j(\mathbf{w}, \mathbf{r}))$, where $f_j()$ is the node function. The evaluation of N() is inherently parallel and the time to evaluate N() on a single input vector is O(#layers). If pipelining is used, multiple input vectors can be evaluated in constant time.

# 3 LEARNING

The "learning" problem for a neural network refers to the problem of finding a network function which approximates some desired "target" function $\mathbf{T}()$, defined over the same set of input vectors as the network function. The problem is simplified by asking that the network function match the target function on only a finite set of input vectors, the "training set" $R$. This is usually done with an error measure. The most common measure is sum-squared error, which we use to define the "instance error" between $\mathbf{N}(\mathbf{w}, \mathbf{r})$ and $\mathbf{T}(\mathbf{r})$ at weight vector $\mathbf{w}$ and input vector $\mathbf{r}$:

$$e_{\mathbf{N},\mathbf{T}}(\mathbf{w}, \mathbf{r}) = \sum_{i \in \text{outputs}} \tfrac{1}{2}(T_i(\mathbf{r}) - o_i(\mathbf{w}, \mathbf{r}))^2 = \tfrac{1}{2}\|\mathbf{T}(\mathbf{r}) - \mathbf{N}(\mathbf{w}, \mathbf{r})\|^2.$$

We can now define the "error function" between N() and T() over $R$ as a function of $\mathbf{w}$:

$$E_{\mathbf{N},\mathbf{T},R}(\mathbf{w}) = \sum_{\mathbf{r} \in R} e_{\mathbf{N},\mathbf{T}}(\mathbf{w}, \mathbf{r}).$$

The learning problem is thus reduced to finding a $\mathbf{w}$ for which $E_{\mathbf{N},\mathbf{T},R}(\mathbf{w})$ is minimized. If this minimum value is zero then the network function approximates the target function exactly on all input vectors in the training set. Henceforth, for notational simplicity we will write $e()$ and $E()$ rather than $e_{\mathbf{N},\mathbf{T}}()$ and $E_{\mathbf{N},\mathbf{T},R}()$.

# 4 OPTIMIZATION TECHNIQUES

As we have framed it here, the learning problem is a classic problem in optimization. More specifically, network learning is a problem of function approximation, where the approximating function is a finite parameter-based system. The goal is to find a set of parameter values which minimizes a cost function, which in this case, is a measure of the error between the target function and the approximating function.

Among the optimization algorithms that can be used to solve this type of problem, gradient-based algorithms have proven to be effective in a variety of applications {Avriel, 1976}. These algorithms are iterative in nature, thus $\mathbf{w}_k$ is the weight vector at the $k_{th}$ iteration. Each iteration is characterized by a *search direction* $\mathbf{d}_k$ and a *step* $\alpha_k$. The weight vector is updated by taking a *step* in the *search direction* as below:

```
for(k=0; evaluate(wk) != CONVERGED; ++k) {
    dk = determine_search_direction();
    αk = determine_step();
    wk+1 = wk + αk dk;
}
```

If $\mathbf{d}_k$ is a direction of descent, such as the negative of the gradient, a sufficiently small step will reduce the value of $E()$. Optimization algorithms vary in the way they determine $\alpha$ and $\mathbf{d}$, but otherwise they are structured as above.

## 5 CONVERGENCE CRITERION

The choice of convergence criterion is important. An algorithm must terminate when $E()$ has been sufficiently minimized. This may be done with a threshold on the value of $E()$, but this alone is not sufficient. In the case where the error surface contains "bad" local minima, it is possible that the error threshold will be unattainable, and in this case the algorithm will never terminate. Some researchers have proposed the use of an iteration limit to guarantee termination despite an unattainable error threshold {Fahlman, 1989}. Unfortunately, for practical problems where this limit is not known *a priori*, this approach is inapplicable.

A necessary condition for $\mathbf{w}^*$ to be a minimum, either local or global, is that the gradient $\mathbf{g}(\mathbf{w}^*) = \nabla E(\mathbf{w}^*) = 0$. Hence, the most usual convergence criterion for optimization algorithms is $\|\mathbf{g}(\mathbf{w}_k)\| \leq \epsilon$ where $\epsilon$ is a sufficiently small *gradient threshold*. The downside of using this as a convergence test is that, for successful trials, learning times will be longer than they would be in the case of an error threshold. Error tolerances are usually specified in terms of an acceptable bit error, and a threshold on the *maximum bit error (MBE)* is a more appropriate representation of this criterion than is a simple error threshold. For this reason we have chosen a convergence criterion consisting of a gradient threshold and an $MBE$ threshold $(\tau)$, terminating when $\|\mathbf{g}(\mathbf{w}_k)\| \leq \epsilon$ or $MBE(\mathbf{w}_k) \leq \tau$, where $MBE()$ is defined as:

$$MBE(\mathbf{w}_k) = \max_{\mathbf{r} \in R} \left( \max_{i \in \text{outputs}} \left( \tfrac{1}{2}(T_i(\mathbf{r}) - o_i(\mathbf{w}_k, \mathbf{r}))^2 \right) \right).$$

## 6 STEEPEST DESCENT

Steepest Descent is the most classical gradient-based optimization algorithm. In this algorithm the search direction $\mathbf{d}_k$ is always the negative of the gradient – the direction of steepest descent. For network learning problems the computation of $\mathbf{g}(\mathbf{w})$, the gradient of $E(\mathbf{w})$, is straightforward:

$$\mathbf{g}(\mathbf{w}) = \nabla E(\mathbf{w}) = \left[ \frac{d}{d\mathbf{w}} \sum_{\mathbf{r} \in R} e(\mathbf{w}, \mathbf{r}) \right]^T = \sum_{\mathbf{r} \in R} \nabla e(\mathbf{w}, \mathbf{r}),$$

where

$$\nabla e(\mathbf{w}, \mathbf{r}) = \left[ \frac{\partial e(\mathbf{w}, \mathbf{r})}{\partial w_{11}}, \frac{\partial e(\mathbf{w}, \mathbf{r})}{\partial w_{12}}, \ldots, \frac{\partial e(\mathbf{w}, \mathbf{r})}{\partial w_{mn}} \right]^T.$$

$$\frac{\partial e(\mathbf{w}, \mathbf{r})}{\partial w_{ij}} = o_i(\mathbf{w}, \mathbf{r}) \delta_j(\mathbf{w}, \mathbf{r}),$$

where for output units

$$\delta_j(\mathbf{w}, \mathbf{r}) = f_j'(i_j(\mathbf{w}, \mathbf{r}))(o_j(\mathbf{w}, \mathbf{r}) - T_j(\mathbf{r})),$$

while for all other units

$$\delta_j(\mathbf{w}, \mathbf{r}) = f_j'(i_j(\mathbf{w}, \mathbf{r})) \sum_{k \in \text{fanout}_j} \delta_j(\mathbf{w}, \mathbf{r}) w_{jk}.$$

The evaluation of $\mathbf{g}$ is thus almost dual to the evaluation of $\mathbf{N}$; while the latter feeds forward through the net, the former feeds back. Both computations are inherently parallelizable and of the same complexity.

The method of Steepest Descent determines the step $\alpha_k$ by *inexact linesearch*, meaning that it minimizes $E(\mathbf{w}_k - \alpha_k \mathbf{d}_k)$. There are many ways to perform this computation, but they are all iterative in nature and thus involve the evaluation of $E(\mathbf{w}_k - \alpha_k \mathbf{d}_k)$ for several values of $\alpha_k$. As each evaluation requires a pass through the entire training set, this is expensive. *Curve fitting* techniques are employed to reduce the number of iterations needed to terminate a linesearch. Again, there are many ways to curve fit . We have employed the method of *false position* and used the *Wolfe Test* to terminate a linesearch {Luenberger, 1986}. In practice we find that the typical linesearch in a network learning problem terminates in 2 or 3 iterations.

# 7 PARTIAL CONJUGATE GRADIENT METHODS

Because linesearch guarantees that $E(\mathbf{w}_{k+1}) < E(\mathbf{w}_k)$, the Steepest Descent algorithm can be proven to converge for a large class of problems {Luenberger, 1986}. Unfortunately, its convergence rate is only linear and it suffers from the problem of "cross-stitching" {Luenberger, 1986}, so it may require a large number of iterations. One way to guarantee a faster convergence rate is to make use of higher order derivatives. Others have investigated the performance of algorithms of this class on network learning tasks, with mixed results {Becker, 1989}. We are not interested in such techniques because they are less parallelizable than the methods we have pursued and because they are more expensive, both computationally and in terms of storage requirements. Because we are implementing our algorithms on the Connection Machine, where memory is extremely limited, this last concern is of special importance. We thus confine our investigation to algorithms that require explicit evaluation only of $\mathbf{g}$, the first derivative.

Conjugate gradient techniques take advantage of second order information to avoid the problem of cross-stitching without requiring the estimation and storage of the Hessian (matrix of second-order partials). The search direction is a combination of the current gradient and the previous search direction:

$$\mathbf{d}_{k+1} = -\mathbf{g}_{k+1} + \beta_k \mathbf{d}_k.$$

There are various rules for determining $\beta_k$; we have had the most success with the Polak-Ribiere rule, where $\beta_k$ is determined from $\mathbf{g}_{k+1}$ and $\mathbf{g}_k$ according to

$$\beta_k = \frac{(\mathbf{g}_{k+1} - \mathbf{g}_k)^T \cdot \mathbf{g}_{k+1}}{\mathbf{g}_k^T \cdot \mathbf{g}_k}.$$

As in the Steepest Descent algorithm, $\alpha_k$ is determined by linesearch. With a simple reinitialization procedure partial conjugate gradient techniques are as robust as the method of Steepest Descent {Powell, 1977}; in practice we find that the Polak-Ribiere method requires far fewer iterations than Steepest Descent.

# 8 BACKPROPAGATION

The Batch Back-propagation algorithm {Rumelhart, 1986} can be described in terms of our optimization framework. Without momentum, the algorithm is very similar to the method of Steepest Descent in that $\mathbf{d}_k = -\mathbf{g}_k$. Rather than being determined by a linesearch, $\alpha$, the "learning rate", is a fixed user-supplied constant. With momentum, the algorithm is similar to a partial conjugate gradient method, as $\mathbf{d}_{k+1} = -\mathbf{g}_{k+1} + \beta_k \mathbf{d}_k$, though again $\beta$, the "momentum term", is fixed. On-line Back-propagation is a variation which makes a change to the weight vector following the presentation of each input vector: $\mathbf{d}_k = \nabla e(\mathbf{w}_k, \mathbf{r}_k)$.

Though very simple, we can see that this algorithm is numerically unsound for several reasons. Because $\beta$ is fixed, $\mathbf{d}_k$ may not be a descent direction, and in this case any $\alpha$ will increase $E()$. Even if $\mathbf{d}_k$ is a direction of descent (as is the case for Batch Back-propagation without momentum), $\alpha$ may be large enough to move from one wall of a "valley" to the opposite wall, again resulting in an increase in $E()$. Because the algorithm can not guarantee that $E()$ is reduced by successive iterations, it cannot be proven to converge. In practice, finding a value for $\alpha$ which results in fast progress and stable behavior is a black art, at best.

# 9 WEIGHT DECAY

One of the problems of performing gradient descent on the "error surface" is that minima may be at infinity. (In fact, for boolean learning problems all minima are at infinity.) Thus an algorithm may have to travel a great distance through weightspace before it converges. Many researchers have found that weight decay is useful for reducing learning times {Hinton, 1986}. This technique can be viewed as adding a term corresponding to the length of the weight vector to the cost function; this modifies the cost surface in a way that bounds all the minima. Rather than minimizing on the error surface, minimization is performed on the surface with cost function

$$C(\mathbf{w}) = E(\mathbf{w}) + \frac{\gamma}{2}\|\mathbf{w}\|^2$$

where $\gamma$, the relative weight cost, is a problem-specific parameter. The gradient for this cost function is $\mathbf{g}(\mathbf{w}) = \nabla C(\mathbf{w}) = \nabla E(\mathbf{w}) + \gamma \mathbf{w}$, and for any step $\alpha_k$, the effect of $\gamma$ is to "decay" the weight vector by a factor of $(1 - \alpha_k \gamma)$:

$$\mathbf{w}_{k+1} = \mathbf{w}_k - \alpha_k \mathbf{g}_k = \mathbf{w}_k(1 - \alpha_k \gamma) - \alpha_k \nabla E(\mathbf{w}_k).$$

# 10 PARALLEL IMPLEMENTATION ISSUES

We have emphasized the parallelism inherent in the evaluation of $E()$ and $\mathbf{g}()$. To be efficient, any learning algorithm must exploit this parallelism. Without momentum, the Back-propagation algorithm is the simplest gradient descent technique, as it requires the storage of only a single vector, $\mathbf{g}_k$. Momentum requires the storage of only one additional vector, $\mathbf{d}_{k-1}$. The Steepest Descent algorithm also requires the storage of only a single vector more than Back-propagation without momentum:

$\mathbf{d}_k$, which is needed for linesearch. In addition to $\mathbf{d}_k$, the Polak-Ribiere method requires the storage of two additional vectors: $\mathbf{d}_{k-1}$ and $\mathbf{g}_{k-1}$. The additional storage requirements of the optimization techniques are thus minimal. The additional computational requirements are essentially those needed for linesearch – a single dot product and a single broadcast per iteration. These operations are parallelizable (log time on the Connection Machine) so the additional computation required by these algorithms is also minimal, especially since computation time is dominated by the evaluation of $E()$ and $\mathbf{g}()$. Both the Steepest Descent and Polak-Ribiere algorithms are easily parallelizable. We have implemented these algorithms, as well as Back-propagation, on a Connection Machine {Hillis, 1986}.

## 11 EXPERIMENTAL RESULTS – BOOLEAN LEARNING

We have compared the performance of the Polak-Ribiere (P-R), Steepest Descent (S-D), and Batch Back-propagation (B-B) algorithms on small boolean learning problems. In all cases we have found the Polak-Ribiere algorithm to be significantly more efficient than the others. All the problems we looked at were based on three-layer networks (1 hidden layer) using the logistic function for all node functions. Initial weight vectors were generated by randomly choosing each component from $(+r, -r)$. $\gamma$ is the relative weight cost, and $\epsilon$ and $\tau$ define the convergence test. Learning times are measured in terms of epochs (sweeps through the training set).

The encoder problem is easily scaled and has no bad local minima (assuming sufficient hidden units: log(#inputs)). All Back-propagation trials used $\alpha = 1$ and $\beta = 0$; these values were found to work about as well as any others. Table 1 summarizes the results. Standard deviations for all data were insignificant ($< 25\%$).

### TABLE 1. Encoder Results

| Encoder Problem | num trials | Parameter Values | | | | Average Epochs to Convergence | | |
|---|---|---|---|---|---|---|---|---|
| | | $r$ | $\gamma$ | $\tau$ | $\epsilon$ | P-R | S-D | B-B |
| 10-5-10 | 100 | 1.0 | 1e-4 | **1e-1** | 1e-8 | 63.71 | 109.06 | 196.93 |
| 10-5-10 | 100 | 1.0 | 1e-4 | **2e-2** | 1e-8 | 71.27 | 142.31 | 299.55 |
| 10-5-10 | 100 | 1.0 | 1e-4 | **7e-4** | 1e-8 | 104.70 | 431.43 | 3286.20 |
| 10-5-10 | 100 | 1.0 | 1e-4 | 0.0 | **1e-4** | 279.52 | 1490.00 | 13117.00 |
| 10-5-10 | 100 | 1.0 | 1e-4 | 0.0 | **1e-6** | 353.30 | 2265.00 | 24910.00 |
| 10-5-10 | 100 | 1.0 | 1e-4 | 0.0 | **1e-8** | 417.90 | 2863.00 | 35260.00 |
| **4-2-4** | 100 | 1.0 | 1e-4 | 0.1 | 1e-8 | 36.92 | 56.90 | 179.95 |
| **8-3-8** | 100 | 1.0 | 1e-4 | 0.1 | 1e-8 | 67.63 | 194.80 | 594.76 |
| **16-4-16** | 100 | 1.0 | 1e-4 | 0.1 | 1e-8 | 121.30 | 572.80 | 990.33 |
| **32-5-32** | 25 | 1.0 | 1e-4 | 0.1 | 1e-8 | 208.60 | 1379.40 | 1826.15 |
| **64-6-64** | 25 | 1.0 | 1e-4 | 0.1 | 1e-8 | 405.60 | 4187.30 | > 10000 |

The parity problem is interesting because it is also easily scaled and its weightspace is known to contain bad local minima. To report learning times for problems with bad local minima, we use *expected epochs to solution, EES*. This measure makes sense especially if one considers an algorithm with a restart procedure: if the algorithm terminates in a bad local minima it can restart from a new random weight vector. *EES* can be estimated from a set of independent learning trials as the ratio of total epochs to successful trials. The results of the parity experiments are summarized in table 2. Again, the optimization techniques were more efficient than Back-propagation. This fact is most evident in the case of bad trials. All trials used $r = 1$, $\gamma = 1e-4$, $\tau = 0.1$ and $\epsilon = 1e-8$. Back-propagation used $\alpha = 1$ and $\beta = 0$.

**TABLE 2. Parity Results**

| Parity | alg | trials | $\%_{succ}$ | $avg_{succ}$ | (s.d.) | $avg_{uns}$ | (s.d.) | EES |
|---|---|---|---|---|---|---|---|---|
| 2-2-1 | P-R | 100 | 72% | 73 | (43) | 232 | (54) | 163 |
| | S-D | 100 | 80% | 95 | (115) | 3077 | (339) | 864 |
| | B-B | 100 | 78% | 684 | (1460) | 47915 | (5505) | 14197 |
| 4-4-1 | P-R | 100 | 61% | 352 | (122) | 453 | (117) | 641 |
| | S-D | 100 | 99% | 2052 | (1753) | 18512 | (-) | 2324 |
| | B-B | 100 | 71% | 8704 | (8339) | 95345 | (11930) | 48430 |
| 8-8-1 | P-R | 16 | 50% | 1716 | (748) | 953 | (355) | 2669 |
| | S-D | 6 | - | >10000 | | >10000 | | >10000 |
| | B-B | 2 | - | >100000 | | >100000 | | >100000 |

# 12 LETTER RECOGNITION

One criticism of batch-based gradient descent techniques is that for large real-world, real-valued learning problems, they will be be less efficient than On-line Back-propagation. The task of characterizing hand drawn examples of the 26 capital letters was chosen as a good problem to test this, partly because others have used this problem to demonstrate that On-line Back-propagation is more efficient than Batch Back-propagation {Le Cun, 1986}. The experimental setup was as follows:

Characters were hand-entered in a 80 × 120 pixel window with a 5 pixel-wide brush (mouse controlled). Because the objective was to have many noisy examples of the same input pattern, not to learn scale and orientation invariance, all characters were roughly centered and roughly the full size of the window. Following character entry, the input window was symbolically gridded to define 100 8 × 12 pixel regions. Each of these regions was an input and the percentage of "on" pixels in the region was its value. There were thus 100 inputs, each of which could have any of 96 (8 × 12) distinct values. 26 outputs were used to represent a one-hot encoding of the 26 letters, and a network with a single hidden layer containing 10 units was chosen. The network thus had a 100-10-26 architecture; all nodes used the logistic function.

A training set consisting of 64 distinct sets of the 26 upper case letters was created by hand in the manner described. 25 "A" vectors are shown in figure 1. This large training set was recursively split in half to define a series of 6 successively larger training sets; $R_0$ to $R_6$, where $R_0$ is the smallest training set consisting of 1 of each letter and $R_i$ contains $R_{i-1}$ and $2^{i-1}$ new letter sets. A testing set consisting of 10 more sets of hand-entered characters was also created to measure network performance. For each $R_i$, we compared naive learning to incremental learning, where naive learning means initializing $w_0^{(i)}$ randomly and incremental learning means setting $w_0^{(i)}$ to $w_*^{(i-1)}$ (the solution weight vector to the learning problem based on $R_{i-1}$). The incremental epoch count for the problem based on $R_i$ was normalized to the number of epochs needed starting from $w_*^{(i-1)}$ plus $\frac{1}{2}$ the number of epochs taken by the problem based on $R_{i-1}$ (since $|R_{i-1}| = \frac{1}{2}|R_i|$). This normalized count thus reflects the total number of relative epochs needed to get from a naive network to a solution incrementally.

Both Polak-Ribiere and On-line Back-propagation were tried on all problems. Table 3 contains only results for the Polak-Ribiere method because no combination of weight-decay and learning rate were found for which Back-propagation could find a solution after 1000 times the number of iterations taken by Polak-Ribiere, although values of $\gamma$ from 0.0 to 0.001 and values for $\alpha$ from 1.0 to 0.001 were tried. All problems had $r = 1$, $\gamma = 0.01$, $\tau = 1e - 8$ and $\epsilon = 0.1$. Only a single trial was done for each problem. Performance on the test set is shown in the last column.

**FIGURE 1. 25 "A"s**

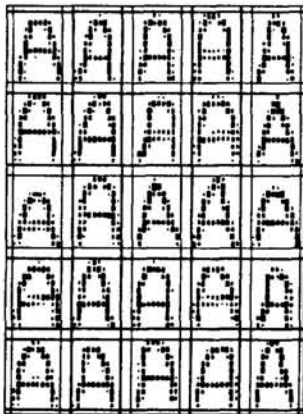

**TABLE 3. Letter Recognition**

| prob set | Learning Time (epochs) | | | Test % |
|---|---|---|---|---|
| | INC | NORM | NAIV | |
| R0 | 95 | 95 | 95 | 53.5 |
| R1 | 83 | 130 | 85 | 69.2 |
| R2 | 63 | 128 | 271 | 80.4 |
| R3 | 14 | 78 | 388 | 83.4 |
| R4 | 191 | 230 | 1129 | 92.3 |
| R5 | 153 | 268 | 1323 | 98.1 |
| R6 | 46 | 180 | 657 | 99.6 |

The incremental learning paradigm was very effective at reducing learning times. Even non-incrementally, the Polak-Ribiere method was more efficient than on-line Back-propagation on this problem. The network with only 10 hidden units was sufficient, indicating that these letters can be encoded by a compact set of features.

## 13 CONCLUSIONS

Describing the computational task of learning in feedforward neural networks as an optimization problem allows exploitation of the wealth of mathematical programming algorithms that have been developed over the years. We have found

that the Polak-Ribiere algorithm offers superior convergence properties and significant speedup over the Back-propagation algorithm. In addition, this algorithm is well-suited to parallel implementation on massively parallel computers such as the Connection Machine. Finally, incremental learning is a way to increase the efficiency of optimization techniques when applied to large real-world learning problems such as that of handwritten character recognition.

## Acknowledgments

The authors would like to thank Greg Sorkin for helpful discussions. This work was supported by the Joint Services Educational Program grant #482427-25304.

## References

{Avriel, 1976} Mordecai Avriel. *Nonlinear Programming, Analysis and Methods.* Prentice-Hall, Inc., Englewood Cliffs, New Jersey, 1976.

{Becker, 1989} Sue Becker and Yan Le Cun. Improving the Convergence of Back-Propagation Learning with Second Order Methods. In *Proceedings of the 1988 Connectionist Models Summer School,* pages 29-37, Morgan Kaufmann, San Mateo Calif., 1989.

{Fahlman, 1989} Scott E. Fahlman. Faster Learning Variations on Back-Propagation: An Empirical Study. In *Proceedings of the 1988 Connectionist Models Summer School,* pages 38-51, Morgan Kaufmann, San Mateo Calif., 1989.

{Hillis, 1986} William D. Hillis. *The Connection Machine.* MIT Press, Cambridge, Mass, 1986.

{Hinton, 1986} G. E. Hinton. Learning Distributed Representations of Concepts. In *Proceedings of the Cognitive Science Society,* pages 1-12, Erlbaum, 1986.

{Kramer, 1989} Alan H. Kramer. Optimization Techniques for Neural Networks. Technical Memo #UCB-ERL-M89-1, U.C. Berkeley Electronics Research Laboratory, Berkeley Calif., Jan. 1989.

{Le Cun, 1986} Yan Le Cun. HLM: A Multilayer Learning Network. In *Proceedings of the 1986 Connectionist Models Summer School,* pages 169-177, Carnegie-Mellon University, Pittsburgh, Penn., 1986.

{Luenberger, 1986} David G. Luenberger. *Linear and Nonlinear Programming.* Addison-Wesley Co., Reading, Mass, 1986.

{Powell, 1977} M. J. D. Powell. "Restart Procedures for the Conjugate Gradient Method", *Mathematical Programming* 12 (1977) 241-254

{Rumelhart, 1986} David E Rumelhart, Geoffrey E. Hinton, and R. J. Williams. Learning Internal Representations by Error Propagation. In *Parallel Distributed Processing: Explorations in the Microstructure of Cognition. Vol 1: Foundations,* pages 318-362, MIT Press, Cambridge, Mass., 1986